# Modeling the spacing effect in sequential category learning

**Hongjing Lu**
Department of Psychology & Statistics
Hongjing@ucla.edu

**Matthew Weiden**
Department of Psychology
mweiden@ucla.edu

**Alan Yuille**
Department of Statistics, Computer Science & Psychology
University of California, Los Angeles
Los Angeles, CA 90095
yuille@stat.ucla.edu

## Abstract

We develop a Bayesian sequential model for category learning. The sequential model updates two category parameters, the mean and the variance, over time. We define conjugate temporal priors to enable closed form solutions to be obtained. This model can be easily extended to supervised and unsupervised learning involving multiple categories. To model the spacing effect, we introduce a generic prior in the temporal updating stage to capture a learning preference, namely, less change for repetition and more change for variation. Finally, we show how this approach can be generalized to efficiently perform model selection to decide whether observations are from one or multiple categories.

## 1 Introduction

Inductive learning  the process by which a new concept or category is acquired through observation of exemplars - poses a fundamental theoretical problem for cognitive science. When exemplars are encountered sequentially, as is typical in everyday learning, then learning is influenced in systematic ways by presentation order. One pervasive phenomenon is the spacing effect, manifested in the finding that given a fixed amount of total study time with a given item, learning is facilitated when presentations of the item are spread across a longer time interval rather than massed into a continuous study period. In category learning, for example, exemplars of two categories can be spaced by presenting them in an interleaved manner (e.g., $A_1B_1A_2B_2A_3B_3$), or massed by presenting them in consecutive blocks (e.g., $A_1A_2A_3B_1B_2B_3$). Kornell & Bjork [1] show that when tested later on classification of novel category members, spaced presentation yields superior performance relative to massed presentation. Similar spacing effects have been obtained in studies of item learning [2] and motor learning [3]. Moreover, spacing effects are found not only in human learning, but also in various types of learning in other species, including rats and Aplysia [4][5].

In the present paper we will focus on spacing effects in the context of sequential category learning. Standard statistical methods based on summary information are unable to deal with order effects, including the performance difference between spaced and massed conditions. From a computational perspective, a sequential learning model is needed to construct category representations from training examples and dynamically update parameters of these representations from trial to trial. Bayesian sequential models have been successfully applied to model causal learning and animal conditioning [6] [7]. In the context of category learning, if we assume that the representation for each category can be specified by a Gaussian distribution where the mean $\mu$ and the variance $\sigma^2$ are both random variables [8], then the learning model must aim to compute the posterior distribution of the parameters for each category given all the observations $x_t$ from trial 1 to trial t, $P(\mu, \sigma^2 | x_t)$.

However, given that both the mean and the variance of a category are random variables, standard Kalman filtering [9] is not directly applicable in this case since it assumes a known variance, which is not warranted in the current application.

In this paper, we extend traditional Kalman filtering in order to update two category parameters, the mean and the variance, over time in the context of category learning. We define conjugate temporal priors to enable closed form solutions to be obtained in this learning model with two unknown parameters. We will illustrate how the learning model can be easily extended to learning situations involving multiple categories either with supervision (i.e., learners are informed of category membership for each training observation) or without supervision (i.e., category membership of each training observation is not provided to learners). Surprisingly, we can also derive closed form solutions in the latter case. This reduces the need for employing particle filters as an approximation to exact inference, commonly used in the case of unsupervised learning [10]. To model the spacing effect, we introduce a generic prior in the temporal updating stage. Finally, we will show how this approach can be generalized to efficiently perform model selection.

The organization of the present paper is as follows. In Section 2 we introduce the Bayesian sequential learning framework in the context of category learning, and discuss the conjugacy property of the model. Section 3 and 4 demonstrate how to develop supervised and unsupervised learning models, which can be compared with human performance. We draw general conclusions in section 5.

## 2 Bayesian sequential model

We adopt the framework of Bayesian sequential learning [11], termed Bayes-Kalman, a probabilistic model in which learning is assumed to be a Markov process with unobserved states. The exemplars in training are directly observable, but the representations of categories are hidden and unobservable. In this paper, we assume that categories can be represented as Gaussian distributions with two unknown parameters, means and variances. These two unknown parameters need to be learned from a limited number of exemplars (e.g., less than ten exemplars).

We now state the general framework and give the update rule for the simplest situation where the training data is generated by a single category specified by a mean $m$ and precision $r$ – the precision is the inverse of the variance and is used to simplify the algebra. Our model assumes that the mean can change over time and is denoted by $m_t$, where $t$ is the time step. The model is specified by the prior distribution $P(m_0, r)$, the likelihood function $P(x|m_t, r)$ for generating the observations, and the temporal prior $P(m_{t+1}|m_t)$ specifying how $m_t$ can vary over time. Note that the precision $r$ is estimated over time, which differs from standard Kalman filtering where it is assumed to be known.

Bayes-Kalman [11] gives iterative equations to determine the posterior $P(m_t, r|X_t)$ after a sequence of observations $X_T = \{x_1, ..., x_t\}$. The update equations are divided into two stages, prediction and correction:

$$P(m_{t+1}, r|X_t) = \int_{-\infty}^{\infty} dm_t P(m_{t+1}|m_t) P(m_t, r|X_t), \qquad (1)$$

$$P(m_{t+1}, r|X_{t+1}) = P(m_{t+1}, r|x_{t+1}, X_t) = \frac{P(x_{t+1}|m_{t+1}, r) P(m_{t+1}, r|X_t)}{P(x_{t+1}|X_t)}. \qquad (2)$$

Intuitively, the Bayes-Kalman first *predicts* the distribution $P(m_{t+1}, r|X_t)$ and then uses this as a prior to *correct* for the new observation $x_{t+1}$ and determine the new posterior $P(m_{t+1}, r|X_{t+1})$. Note that the temporal prior $P(m_{t+1}|m_t)$ implies that the model automatically pays most attention to recent data and does not memorize the data, thus exhibiting sensitivity to the data ordering.

### 2.1 Conjugate priors

The distributions $P(m_0, r), P(x|m_t, r), P(m_{t+1}|m_t)$ are chosen to be conjugate, so that the distribution $P(m_t, r|X_t)$ takes the same functional form as $P(m_0, r)$. As shown in the following section, this reduces the Bayes-Kalman equations to closed form update rules for the parameters of the dis-

tributions. The distributions are specified in terms of Gamma and Gaussian distributions:

$$g(r : \alpha, \beta) = \frac{\beta^\alpha}{\Gamma(\alpha)} r^{\alpha-1} \exp\{-\beta r\}, \ r \geq 0. \text{ Gamma.} \tag{3}$$

$$G(x : \mu, \rho) = \{\frac{\rho}{2\pi}\} \exp\{-\rho/2(x - \mu)^2\}. \text{ Gaussian.} \tag{4}$$

We specify the prior $P(m_0, r)$ as the product of a Gaussian $P(m_0|r)$ and a Gamma $P(r)$:

$$P(m_0|r) = G(m_0 : \mu, \tau r), \qquad P(r) = g(r : \alpha, \beta), \tag{5}$$

where $\mu, \tau, \alpha, \beta$ are the parameters of the distribution. For simplicity, we call this a *Gamma-Gaussian* distribution with parameters $\mu, \tau, \alpha, \beta$.

The likelihood function and temporal prior are both Gaussians:

$$P(x_t|m_t, r) = G(x_t : m_t, \zeta r), \qquad P(m_{t+1}|m_t) = G(m_{t+1} : m_t, \gamma r), \tag{6}$$

where $\zeta, \gamma$ are constants.

The conjugacy of the distributions ensures that the posterior distribution $P(m_t, r|X_t)$ will also be a Gamma-Gaussian distribution with parameters $\mu_t, \tau_t, \alpha_t, \beta_t$, where the update rules for these parameters are specified in the next section.

## 2.2 Update rules for the model parameters

The update rules for the model parameters follow from substituting the distributions into the Bayes-Kalman equations 1, 2. We sketch how these update rules are obtained assuming that $P(m_t, r|X_t)$ is a Gamma-Gaussian with parameters $\mu_t, \tau_t, \alpha_t, \beta_t$, which is true for $t = 0$ using equations (5,6).

The form of the prediction equation and the temporal prior, see equations (1,6), ensures that $P(m_{t+1}, r|X_t)$ is also a Gamma-Gaussian distribution with parameters $\mu_t, \tau_t^p, \alpha_t, \beta_t$, where

$$\tau_t^p = \frac{\tau_t \gamma}{\tau_t + \gamma}. \tag{7}$$

The correction equation and the likelihood function, see equations (2,6), ensure that $P(m_{t+1}, r|X_{t+1})$ is also Gamma-Gaussian with parameters $\mu_{t+1}, \tau_{t+1}, \alpha_{t+1}, \beta_{t+1}$ given by:

$$\alpha_{t+1} = \alpha_t + 1/2, \qquad \beta_{t+1} = \beta_t + \frac{\zeta \tau_t^p (x_{t+1} - \mu_t)^2}{2(\zeta + \tau_t^p)},$$

$$\mu_{t+1} = \frac{\zeta x_{t+1} + \tau_t^p \mu_t}{\zeta + \tau_t^p}, \qquad\qquad \tau_{t+1} = \zeta + \tau_t^p. \tag{8}$$

Intuitively, the prediction only reduces the precision of $m$ but makes no change to its mean or to the distribution over $r$. By contrast, the new observation alters the mean of $m$ (moving it closer to the new observation $x_{t+1}$), and also increases its precision, which sharpens the distribution on $r$.

## 2.3 Model evidence

We also need to compute the probability of the observation sequence $X_t$ from the model (which will be used later for model selection). This can be expressed recursively as:

$$p(X_t) = p(x_t|X_{t-1})p(x_{t-1}|X_{t-2})...p(x_1). \tag{9}$$

This computation is also simplified because we use conjugate distributions. The terms in equation (9) can be expressed as $P(x_{t+1}|X_t) = \int dm_{t+1} dr P(x_{t+1}|m_{t+1}, r) P(m_{t+1}, r|X_t)$ and these integrals can be calculated analytically yielding:

$$P(x_{t+1}|X_t) = \{\beta_t + \frac{\zeta \tau_t}{2(\zeta + \tau_t)}(x - \mu_t)^2\}^{-(\alpha_t + 1/2)} \{\frac{1}{2\pi} \frac{\zeta \tau_t}{\zeta + \tau_t}\}^{1/2} \frac{\beta_t^{\alpha_t} \Gamma(\alpha_t + 1/2)}{\Gamma(\alpha_t)}. \tag{10}$$

# 3 Supervised category learning

Although the learning model is presented for one category, it can easily be extended to learning multiple categories with known category membership for training data (i.e., under supervision). In this section, we will first describe an experiment with two categories to show how the category representations change over time; then we will simulate learning with six categories and compare predictions with human data in psychological experiments.

## 3.1 Two-category learning with supervision

We first conduct a synthetic experiment with two categories under supervision. We generate six training observations from one of two one-dimensional Gaussian distributions (representing categories A and B, respectively) with means $[-0.4, 0.4]$ and standard deviation of 0.4. Two training conditions are included, a massed condition with the data presentation order of AAABBB and a spaced condition with the order of ABABAB.

To model the acquisition of category representations during training, we employ the Bayesian learning model as described in the previous section. In the correction stage of each trial, the model updates the parameters corresponding to the category that produced the observation based on the supervision (i.e., known category membership), following equation (8).

In the prediction stage, however, different values of a fixed model parameter $\gamma$ are introduced to incorporate a generic prior that controls how much the learner is willing to update category representations from one trial to the next. The basic hypothesis is that learners will have greater confidence in knowledge of a category presented on trial $t$ than of a category absent on trial $t$. As a consequence, the learner will be willing to accept more change in a category representation if the observation on the previous trial was drawn from a different category. This generic prior does share some conceptual similarity with a model developed by Kording et. al,[?], which assumes that the moment-moment variance of the states is higher for faster timescales (p. 779).

More specifically, if the observation on trial $t$ is from the first category, in the prediction phase we will update the $\tau_t$ parameters for the two categories, $\tau_t{}^1, \tau_t{}^2$, as:

$$\tau_t{}^1 \mapsto \frac{\tau_t{}^1 \gamma_s}{\tau_t{}^1 + \gamma_s}, \qquad \tau_t{}^2 \mapsto \frac{\tau_t{}^2 \gamma_d}{\tau_t{}^2 + \gamma_d}, \tag{11}$$

in which $\gamma_s > \gamma_d$. In the simulation, we used $\gamma_s = 50$ and $\gamma_d = .5$

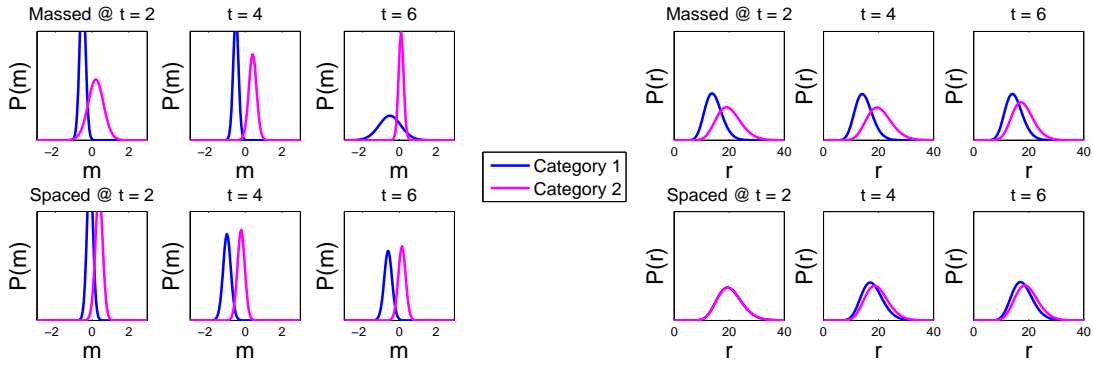

Figure 1: Posterior distributions of means $P(m_t|X_t)$ and precisions $P(r_t|X_t)$ updated on training trials in two-category supervised learning. Blue lines indicate category parameters in the first category; and red lines indicate parameters in the second category. The top panel shows the results for the massed condition (i.e., AAABBB), and the bottom panel shows the results for the spaced condition (i.e., ABABAB). Please see in colour. We show the distributions only on even trials to save space. See section 3.1.

Figure (1) shows the change of posterior distributions of the two unknown category parameters, means $P(m_t|X_t)$ and precisions $P(r_t|X_t)$, over training trials. Figure (2) shows the category representation in the form of the posterior distribution of $P(x_t|X_t)$. In the massed condition (i.e.,

AAABBB), the variance of the first category decreases over the first three trials, and then increases over the second three trials because the observations are from the second category. The increase of category variance reflects the forgetting that occurs if no new observations are provided for a particular category after a long interval. This type of forgetting does not occur in the spaced condition, as the interleaved presentation order ABABAB ensured that each category recurs after a short interval.

Based upon the learned category representations, we can compute accuracy (the ability to discriminate between the two learnt distributions) using the posterior distributions of the two categories. After 100 simulations, the average accuracy in the massed condition is 0.78, which is lower than the 0.84 accuracy in the spaced condition. Thus our model is able to predict the spacing effect found in two-category supervised learning.

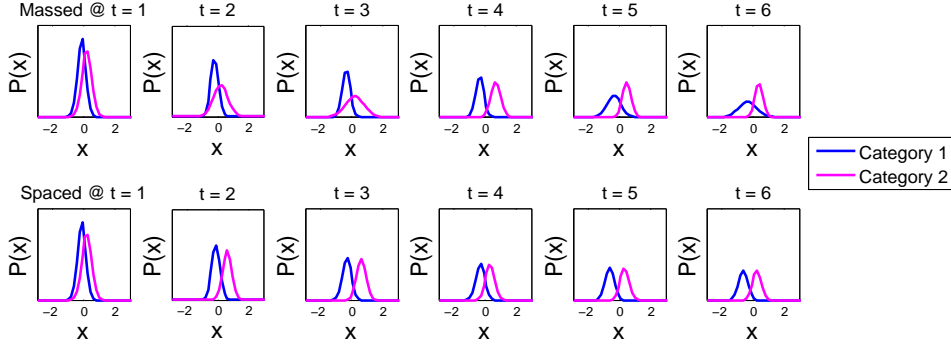

Figure 2: Posterior distribution of each category, $P(x_t|X_t)$, updated on training trials in the two-category supervised learning. Same conventions as in figure (1). See section 3.1.

## 3.2   Modeling the spacing effect in six-category learning

Kornell and Bjork [1] asked human subjects to study six paintings by six different artists, with a given artists paintings presented consecutively (massed) or interleaved with other artists paintings (spaced). In the training phase, subjects were informed which artist created each training painting. The same 36 paintings were studied in the training phase, but with different presentation orders in the massed and spaced conditions. In the subsequent test phase, six new paintings (one from each artist) were presented and subjects had to identify which artist painted each of a series of new paintings. Four test blocks were tested with random display order for artists. In each test block, participants were given feedback after making an identification response. Paintings presented in one test block thus served as training examples for the subsequent test block. Human results are shown in figure (4). Human subjects showed significantly better test performance after spaced than massed training. Given that feedback was provided and one painting from each artist appeared in one test block, it is not surprising that test performance increased across test blocks and the spacing effect decreased with more test blocks.

To simulate the data, we generated training and test data from six one-dimensional Gaussian distributions with means $[-2, -1.2, -0.4, 0.4, 1.2, 2]$ and standard deviation of 0.4. Figure (3) shows the learned category representations in terms of posterior distributions. Depending on the presentation order of training data (massed or spaced), the learned distributions differ in terms of means and variances for each category. To compare with human performance reported by Kornell and Bjork, the model estimates accuracy in terms of discrimination between the two categories based upon learned distributions. Figure (4) shows average accuracy from 1000 simulations. The result plot illustrates that the model predictions match human performance well.

## 4   Unsupervised category learning

Both humans and animals can learn without supervision. For example, in the animal conditioning literature, various studies have shown that exposing two stimuli in blocks (equivalent to a massed condition) is less effective in producing generalization [12]. Balleine et. al. [4] found that with rats, preexposure to two stimuli A and B (massed or spaced) determines the degree to which backward blocking is subsequently obtained – backward blocking occurs if the preexposure is spaced but not

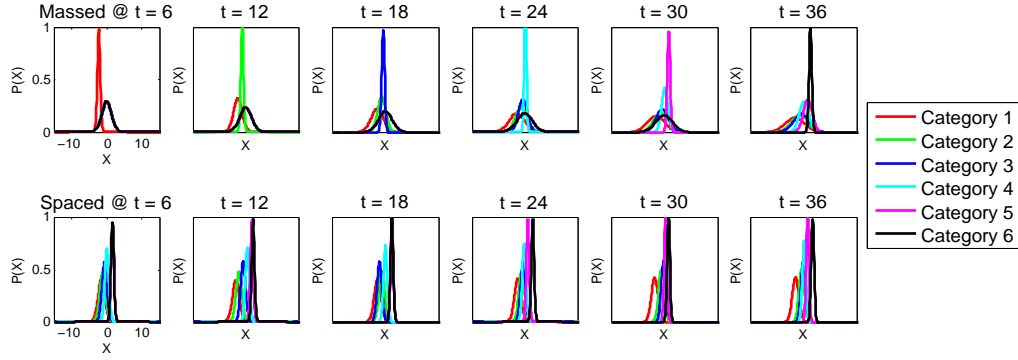

Figure 3: Posterior distribution of each category, $P(x_t|X_t)$, updated on training trials in the six-category supervised learning. Same conventions as in figure (1). See section 3.2.

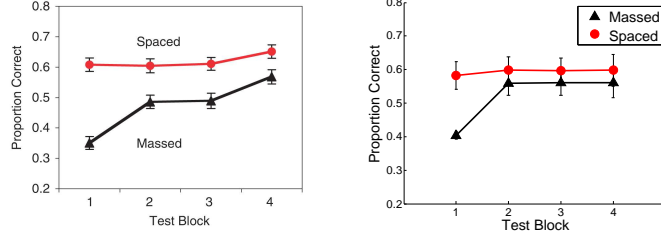

Figure 4: Human performance (left) and model prediction (right). Proportion correct as a function of presentation training conditions (massed and spaced) and test block. See section 3.2.

if it is massed. They conclude that in the massed preexposure the rats are unable to distinguish two separate categories for A and B, and therefore treat them as members of a single category. By contrast, they conclude that rats can distinguish the categories A and B in the spaced preexposure.

In this section, we generalize the sequential category model to unsupervised learning, when the category membership of each training example is not provided to observers. We first derive the extension of the sequential model to this case (surprisingly, showing we can obtain all results in closed form). Then we determine whether massed and spaced stimuli (as in Balleine et. al.'s experiment [4]) are most likely to have been generated by a single category or by two categories. We also assess the importance of supervision in training by comparing performance after unsupervised learning with that after supervised learning.

We consider a model with two hidden categories. Each category can be represented as a Gaussian distribution with a mean and precision $m^1, r^1$ and $m^2, r^2$. The likelihood function assumes that the data is generated by either category with equal probability, since the category membership is not provided,

$$P(x|m^1, r^1, m^2, r^2) = \frac{1}{2}P(x|m^1, r^1) + \frac{1}{2}P(x|m^2, r^2), \qquad (12)$$

$$\text{with } P(x|m^1, r^1) = G(x : m^1, \zeta r^1), \quad P(x|m^2, r^2) = G(x : m^2, \zeta r^2). \qquad (13)$$

We specify prior distributions and temporal priors as before:

$$P(m_0^1, r^1) = G(m_0^1 : \mu^1, \tau r^1), \qquad P(m_0^2, r^2) = G(m_0^2 : \mu^2, \tau r^2) \qquad (14)$$

$$P(m_{t+1}^1|m_t^1) = G(m_{t+1}^1 : m_t^1, \gamma r^1), \qquad P(m_{t+1}^2|m_t^2) = G(m_{t+1}^2 : m_t^2, \gamma r^2). \qquad (15)$$

The joint posterior distribution $P(m_t^1, r^1, m_t^2, r^2|X_t)$ after observations $X_t$ can be formally obtained by applying the Bayes-Kalman update rules to the joint distribution – i.e., replace $(m_t, r)$ by $(m_t^1, r^1, m_t^2, r^2)$ in equations (1,2)). But this update is more complicated because we do not know whether the new observation $x_t$ should be assigned to category 1 or category 2. Instead we have to sum over all the possible assignments of the observations to the categories which gives $2^t$ possible assignments at time $t$. This can be performed efficiently in a recursive manner. Let $A_t$ denote the set of possible assignments at time $t$ where each assignment is a string $(a_1, ..., a_t)$ of binary variables

of length $t$, where $(1, ..., 1)$ is the assignment where all the observations are assigned to category 1, $(2, 1, ..., 1)$ assigns the first observation to category 2 and the remainder to category 1, and so on.

By substituting equations (12,14,15) into Bayes-Kalman we can obtain an iterative update equation for $P(m_t^1, r^1, m_t^2, r^2 | X_t)$. At time $t$ we represent:

$$P(m_t^1, r^1, m_t^2, r^2 | X_t) = \sum_{(a_1,...,a_t) \in A_t} P(m^1, r | \vec{\alpha}_{a_1,...,a_t}^1) P(m^2, r | \vec{\alpha}_{(a_1,...,a_t)}^2) P(a_1, ..., a_t | X_t), \tag{16}$$

where $\alpha_{(a_1,...,a_t)}^i$ denotes the values of the parameters $\vec{\alpha} = (\alpha, \beta, \mu, \tau)$ for category $i$ ($i \in \{1, 2\}$) for observation sequence $(a_1, ..., a_t)$ and $P(a_1, ..., a_t)$ is the probability of assignment $(a_1, ..., a_t)$. At $t = 0$ there is no observation sequence and $P(m_0^1, r^1, m_0^2, r^2 | X_t) = P(m^1, r | \vec{\alpha}^1) P(m^2, r | \vec{\alpha}^2)$ which corresponds to $A_0$ containing a single element which has probability one.

The prediction stage updates the $\tau$ component of $\vec{\alpha}^i(a_1, ..., a_t)$ by:

$$\tau^i(a_1, ..., a_t) \mapsto \frac{\gamma^i(a_t) \tau^i(a_1, ..., a_t)}{\gamma^i(a_t) + \tau^i(a_1, ..., a_t)}. \tag{17}$$

We define $\gamma^i(a_t)$ as larger if $i = a_t$ and smaller if $i \neq a_t$, as specified in equation (11) to incorporate the generic prior described in section 3.1.

The correction stage at time $t + 1$ introduces another observation, which must be assigned to the two categories. This gives a new set $A_{t+1}$ of $2^{t+1}$ assignments of form $(a_1, ..., a_{t+1})$ and a new posterior:

$$P(m_{t+1}^1, r^1, m_{t+1}^2, r^2 | X_{t+1}) = \sum_{(a_1,...,a_{t+1}) \in A_{t+1}} P(m^1, r | \vec{\alpha}_{a_1,...,a_{t+1}}^1) P(m^2, r | \vec{\alpha}_{(a_1,...,a_{t+1})}^2) P(a_1, ..., a_{t+1} | X_{t+1}), \tag{18}$$

where we compute $\vec{\alpha}_{(a_1,...,a_{t+1})}^i$ for $i \in \{1, 2\}$ by:

$$\alpha_{(a_1,...,a_{t+1})}^i = \alpha_{(a_1,...,a_t)}^i + 1/2, \quad \beta_{(a_1,...,a_{t+1})}^1 = \beta_{(a_1,...,a_t)}^1 + \frac{\zeta \tau_{(a_1,...,a_t)}^i (x_{t+1} - \mu_{(a_1,...,a_t)}^1)^2}{2(\zeta + \tau_{(a_1,...,a_t)}^i)},$$

$$\mu_{(a_1,...,a_{t+1})}^i = \frac{\zeta x_{t+1} + \tau_{(a_1,...,a_t)}^i \mu_{(a_1,...,a_t)}^i}{\zeta + \tau_{(a_1,...,a_t)}^1}, \qquad \tau_{(a_1,...,a_{t+1})}^i = \zeta + \tau_{(a_1,...,a_t)}^i, \tag{19}$$

and we compute $P(a_1, ..., a_{t+1})$ by:

$$P(a_1, ..., a_{t+1} | X_{t+1}) = \frac{P(x_{t+1} | \vec{\alpha}_{(a_1,...,a_t)}^{a_{t+1}}) P(a_1, ..., a_t)}{\sum_{(a_1,...,a_t)} P(x_{t+1} | \vec{\alpha}_{(a_1,...,a_t)}^{a_{t+1}}) P(a_1, ..., a_t)}, \tag{20}$$

where

$$P(x_{t+1} | \vec{\alpha}_{(a_1,...,a_t)}^{a_{t+1}}) = \int dm^{a_{t+1}} dr^{a_{t+1}} P(x_{t+a} | m^{(a_{t+1})}, r^{a_{t+1}}) P(m^{(a_{t+1})}, r^{a_{t+1}} | \vec{\alpha}_{(a_1,...,a_t)}) \tag{21}$$

The model selection can, as before, be expressed as $P(x_t | X_{t-1}) P(x_{t-1} | X_t) .... P(x_1)$, where

$$P(x_{t+1} | X_t) = \sum_{(a_1,...,a_t) \in A_t} P(x_{t+1} | \vec{\alpha}_{(a_1,...,a_t)}^{a_{t+1}}) P(a_1, ..., a_t). \tag{22}$$

We can now address the problem posed by Balleine et. al.'s preexposure experiments [4] – why do rats identify a single category for the massed stimuli but two categories for the spaced stimuli?

We treat this as a model selection problem. We compare the evidence for the sequential model with one category, see equations (9,10), versus the evidence for the model with two categories, see equations (9,22), for the two cases AAABBB (massed) and ABABAB (spaced).

We use the same data as described in section (3.1) but without providing category membership for any of the training data. The left plot in figure (5) shows the result obtained by comparing model evidence for the one-category model with model evidence for the two-category model. A greater ratio value indicates greater support for the one-category account. As shown in figure (5), the model decides that all training observations are from one category in the massed condition, but from two different categories in the spaced condition (using zero as the decision threshold). These predictions agree with with Balleine et. al.'s findings.

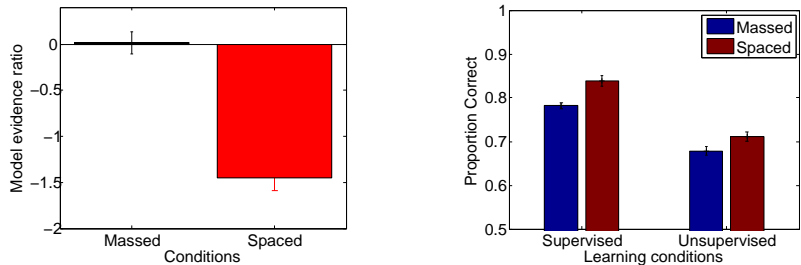

Figure 5: Model selection and accuracy results. Left, model selection results as a function of presentation training conditions (massed and spaced). A greater ratio indicates more support for the one-category account. Error bars indicate the standard error from 100 simulations. See section 4.2. Right, comparison of supervised and unsupervised learning in terms of accuracy. See section 4.3.
To assess the influence of supervision on learning, we compare performance between supervised learning (described in section (3.1)) with unsupervised learning (described in this section). To make the comparison, we assume that learners are provided with the same training data and are informed that the data are from two different categories, either with known category membership (supervised) or unknown category membership (unsupervised) for each training observation. Accuracy measured by discrimination between the two categories is compared in the right plot of figure (5). The model predicts higher accuracy given supervised than unsupervised learning. Furthermore, the model predicts a spacing effect for both types of learning, although the effect is reduced with unsupervised learning.

## 5   Conclusions

In this paper, we develop a Bayesian sequential model for category learning by updating category representations over time based on two category parameters, the mean and the variance. Analytic updating rules are obtained by defining conjugate temporal priors to enable closed form solutions. A generic prior in the temporal updating stage is introduced to model the spacing effect. Parameter estimation and model selection can be performed on the basis of updating rules. The current work extends standard Kalman filtering, and is able to predict learning phenomena that have been observed for humans and other animals.

In addition to explaining the spacing effect, our model predicts that subjects will become less certain about their knowledge of learned categories as time passes, see the increase in category variance in Figure 2. But our model is not standard Kalman filter (since the measurement variance is unknown), so we do not predict exponential decay. Instead, as shown in Equation 10, our model predicts the pattern of power-law forgetting that is fairly universal in human memory [14]

For small number of observations, our model is extremely efficient because we can derive analytic solutions. For example, the analytic solutions for unsupervised learning requires only  0.2 seconds for six observations while numerical integration takes  18 minutes. However, our model will scale exponentially with the number of observations in unsupervised learning. Future work is to include a pruning strategy to keep the complexity practical.

**Acknowledgement**

This research was supported a grant from Air Force FA 9550-08-1-0489.

# References

[1] Kornell, N., & Bjork, R. A. (2008a). Learning concepts and categories: Is spacing the "enemy of induction"? Psychological Science, 19, 585-592.

[2] Bahrick, H.P., Bahrick, L.E., Bahrick, A.S., & Bahrick, P.E. (1993). Maintenance of foreign language vocabulary and the spacing effect. Psychological Science, 4, 316321.

[3] Shea, J.B., & Morgan, R.L. (1979). Contextual interference effects on the acquisition, retention, and transfer of a motor skill. Journal of Experimental Psychology: Human Learning and Memory, 5, 179187.

[4] Balleine, B. W. Espinet, A. & Gonzalez, F. (2005).Perceptual learning enhances retrospective revaluation of conditioned flavor preferences in rats. Journal of Experimental Psychology: Animal Behavior Processes, 31(3): 341-50.

[5] Carew, T.J., Pinsker, H.M., & Kandel, E.R. (1972). Long-term habituation of a defensive withdrawal reflex in Aplysia. Science, 175, 451454.

[6] Daw, N., Courville, A. C, & Dayan, P. (2007). Semi-rational Models of Conditioning: The Case of Trial Order. In M. Oaksford and N. Chater (Eds.). *The probabilistic mind: Prospects for rational models of cognition.* Oxford: Oxford University Press.

[7] Dayan, P. & Kakade, S. (2000). Explaining away in weight space. In T. K. Leen et al., (Eds.), *Advances in neural information processing systems* (Vol. 13, pp. 451-457). Cambridge, MA: MIT Press.

[8] Fried, L. S., & Holyoak, K. J. (1984). Induction of category distributions: A framework for classification learning. Journal of Experimental Psychology: Learning, Memory and Cognition, 10, 234-257.

[9] Kalman, R. E. (1960). A new approach to linear filtering and prediction problems. Transactions of the ASME-Journal of Basic Engineering, 82:35-45.

[10] Schubert, J., & Sidenbladh, H. (2005). Sequential clustering with particle filters: Extimating the number of clusters from data. 7th International Conference on Information Fusion (FUSION).

[11] Ho, Y-C & Lee, R.C.K. (1964). A Bayesian appraoch to problems in stochastic estimation and control. IEEE Transactions on Automatic Control, 9, 333-339.

[12] Honey, R. C., Bateson, P., & Horn, G. (1994). The role of stimulus comparison in perceptual learning: An investigation with the domestic chick. Quarterly Journal of Experimental Psychology: Comparative and Physiological Psychology, 47(B), 83103.

[13] Kording, K. P., Tenenbaum, J. B., and Shadmehr, R. (2007). The dynamics of memory as a consequence of optimal adaptation to a changing body. Nature Neuroscience, 10:779-786.

[14] Anderson, J. R. And Schooler, L. J. (1991). Reflections of the environment in memory. Psychological Science, 2, 395-408.

